# Learning Concept Graphs from Text with Stick-Breaking Priors

**America L. Chambers**
Department of Computer Science
University of California, Irvine
Irvine, CA 92697
ahollowa@ics.uci.edu

**Padhraic Smyth**
Department of Computer Science
University of California, Irvine
Irvine, CA 92607
smyth@ics.uci.edu

**Mark Steyvers**
Department of Cognitive Science
University of California, Irvine
Irvine, CA 92697
mark.steyvers@uci.edu

## Abstract

We present a generative probabilistic model for learning general graph structures, which we term concept graphs, from text. Concept graphs provide a visual summary of the thematic content of a collection of documents—a task that is difficult to accomplish using only keyword search. The proposed model can learn different types of concept graph structures and is capable of utilizing partial prior knowledge about graph structure as well as labeled documents. We describe a generative model that is based on a stick-breaking process for graphs, and a Markov Chain Monte Carlo inference procedure. Experiments on simulated data show that the model can recover known graph structure when learning in both unsupervised and semi-supervised modes. We also show that the proposed model is competitive in terms of empirical log likelihood with existing structure-based topic models (hPAM and hLDA) on real-world text data sets. Finally, we illustrate the application of the model to the problem of updating Wikipedia category graphs.

## 1 Introduction

We present a generative probabilistic model for learning concept graphs from text. We define a concept graph as a rooted, directed graph where the nodes represent thematic units (called concepts) and the edges represent relationships between concepts. Concept graphs are useful for summarizing document collections and providing a visualization of the thematic content and structure of large document sets - a task that is difficult to accomplish using only keyword search. An example of a concept graph is Wikipedia's category graph[1]. Figure 1 shows a small portion of the Wikipedia category graph rooted at the category MACHINE_LEARNING[2]. From the graph we can quickly infer that the collection of machine learning articles in Wikipedia focuses primarily on evolutionary algorithms and Markov models with less emphasis on other aspects of machine learning such as Bayesian networks and kernel methods.

The problem we address in this paper is that of learning a concept graph given a collection of documents where (optionally) we may have concept labels for the documents and an initial graph structure. In the latter scenario, the task is to identify additional concepts in the corpus that are

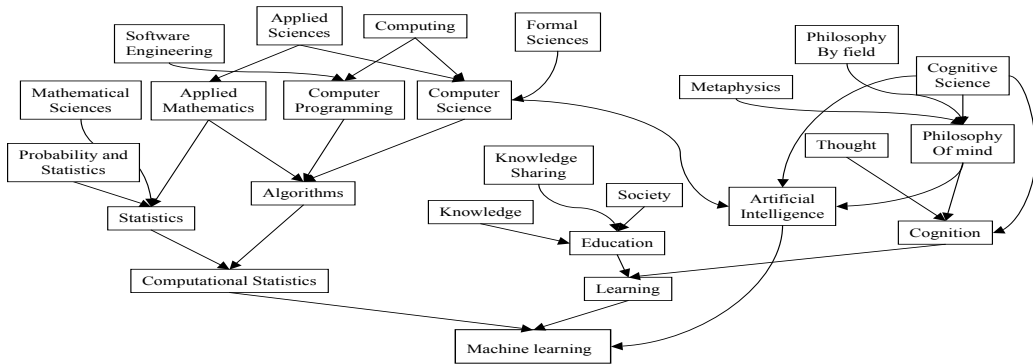

Figure 1: A portion of the Wikipedia category supergraph for the node MACHINE_LEARNING

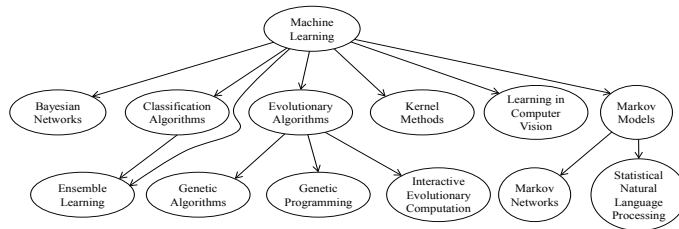

Figure 2: A portion of the Wikipedia category subgraph rooted at the node MACHINE_LEARNING

not reflected in the graph or additional relationships between concepts in the corpus (via the co-occurrence of concepts in documents) that are not reflected in the graph. This is particularly suited for document collections like Wikipedia where the set of articles is changing at such a fast rate that an automatic method for updating the concept graph may be preferable to manual editing or re-learning the hierachy from scratch. The foundation of our approach is latent Dirichlet allocation (LDA) [1]. LDA is a probabilistic model for automatically identifying topics within a document collection where a topic is a probability distribution over words. The standard LDA model does not include any notion of relationships, or dependence, between topics. In contrast, methods such as the hierarchical topic model (hLDA) [2] learn a set of topics in the form of a tree structure. The restriction to tree structures however is not well suited for large document collections like Wikipedia. Figure 1 gives an example of the highly non-tree like nature of the Wikipedia category graph. The hierarchical Pachinko allocation model (hPAM) [3] is able to learn a set of topics arranged in a fixed-sized graph with a nonparametric version introduced in [4]. The model we propose in this paper is a simpler alternative to hPAM and nonparametric hPAM that can achieve the same flexibility (i.e. learning arbitrary directed acyclic graphs over a possibly infinite number of nodes) within a simpler probabilistic framework. In addition, our model provides a formal mechanism for utilizing labeled data and existing concept graph structures. Other methods for creating concept graphs include the use of techniques such as hierarchical clustering, pattern mining and formal concept analysis to construct ontologies from document collections [5, 6, 7]. Our approach differs in that we utilize a probabilistic framework which enables us (for example) to make inferences about concepts and documents. Our primary novel contribution is the introduction of a flexible probabilistic framework for learning general graph structures from text that is capable of utilizing both unlabeled documents as well as labeled documents and prior knowledge in the form of existing graph structures.

In the next section we introduce the stick-breaking distribution and show how it can be used as a prior for graph structures. We then introduce our generative model and explain how it can be adapted for the case where we have an initial graph structure. We derive collapsed Gibbs' sampling equations for our model and present a series of experiments on simulated and real text data. We compare our performance against hLDA and hPAM as baselines. We conclude with a discussion of the merits and limitations of our approach.

## 2 Stick-breaking Distributions

Stick-breaking distributions $\mathcal{P}(\cdot)$ are discrete probability distributions of the form:

$$\mathcal{P}(\cdot) = \sum_{j=1}^{\infty} \pi_j \delta_{x_j}(\cdot) \quad \text{where} \quad \sum_{j=1}^{\infty} \pi_j = 1, \ 0 \le \pi_j \le 1$$

and $\delta_{x_j}(\cdot)$ is the delta function centered at the atom $x_j$. The $x_j$ variables are sampled independently from a base distribution $H$ (where $H$ is assumed to be continuous). The stick-breaking weights $\pi_j$ have the form

$$\pi_1 = v_1, \quad \pi_j = v_j \prod_{k=1}^{j-1}(1 - v_k) \quad \text{for } j = 2, 3, \ldots, \infty$$

where the $v_j$ are independent Beta$(\alpha_j, \beta_j)$ random variables. Stick-breaking distributions derive their name from the analogy of repeatedly breaking the remainder of a unit-length stick at a randomly chosen breakpoint. See [8] for more details.

Unlike the Chinese restaurant process, the stick-breaking process lacks exchangeability. The probability of sampling a particular cluster from $\mathcal{P}(\cdot)$ given the sequences $\{x_j\}$ and $\{v_j\}$ is *not* equal to the probability of sampling the same cluster given a permutation of the sequences $\{x_{\sigma(j)}\}$ and $\{v_{\sigma(j)}\}$. This can be seen in Equation 2 where the probability of sampling $x_j$ depends upon the value of the $j - 1$ proceeding Beta random variables $\{v_1, v_2, \ldots, v_{j-1}\}$. If we fix $x_j$ and permute every other atom, then the probability of sampling $x_j$ changes: it is now determined by the Beta random variables $\{v_{\sigma(1)}, v_{\sigma(2)}, \ldots, v_{\sigma(j-1)}\}$.

The stick-breaking distribution can be utilized as a prior distribution on graph structures. We construct a prior on graph structures by specifying a distribution at each node (denoted as $\mathcal{P}_t$) that governs the probability of transitioning from node $t$ to another node in the graph. There is some freedom in choosing $\mathcal{P}_t$; however we have two constraints. First, making a new transition must have non-zero probability. In Figure 1 it is clear that from MACHINE_LEARNING we should be able to transition to any of its children. However we may discover evidence for passing directly to a leaf node such as STATISTICAL_NATURAL_LANGUAGE_PROCESSING (e.g. if we observe new articles related to statistical natural language processing that do not use Markov models). Second, making a transition to a new node must have non-zero probability. For example, we may observe new articles related to the topic of Bioinformatics. In this case, we want to add a new node to the graph (BIOINFORMATICS) and assign some probability of transitioning to it from other nodes.

With these two requirements we can now provide a formal definition for $\mathcal{P}_t$. We begin with an initial graph structure $G_0$ with $t = 1 \ldots T$ nodes. For each node $t$ we define a feasible set $\mathcal{F}_t$ as the collection of nodes to which $t$ can transition. The feasible set may contain the children of node $t$ or possible child nodes of node $t$ (as discussed above). In general, $\mathcal{F}_t$ is some subset of the nodes in $G_0$. We add a special node called the "exit node" to $\mathcal{F}_t$. If we sample the exit node then we exit from the graph instead of transitioning forward. We define $\mathcal{P}_t$ as a stick-breaking distribution over the finite set of nodes $\mathcal{F}_t$ where the remaining probability mass is assigned to an infinite set of new nodes (nodes that exist but have not yet been observed). The exact form of $\mathcal{P}_t$ is shown below.

$$\mathcal{P}_t(\cdot) = \sum_{j=1}^{|\mathcal{F}_t|} \pi_{tj} \delta_{f_{tj}}(\cdot) \ + \sum_{j=|\mathcal{F}_t|+1}^{\infty} \pi_{tj} \delta_{x_{tj}}(\cdot)$$

The first $|\mathcal{F}_t|$ atoms of the stick-breaking distribution are the feasible nodes $f_{tj} \in \mathcal{F}_t$. The remaining atoms are unidentifiable nodes that have yet to be observed (denoted as $x_{tj}$ for simplicity).

This is not yet a working definition unless we explicitly state which nodes are in the set $\mathcal{F}_t$. Our model does not in general assume any specific form for $\mathcal{F}_t$. Instead, the user is free to define it as they like. In our experiments, we first assign each node to a unique depth and then define $\mathcal{F}_t$ as any node at the next lower depth. The choice of $\mathcal{F}_t$ determines the type of graph structures that can be learned. For the choice of $\mathcal{F}_t$ used in this paper, edges that traverse multiple depths are not allowed and edges between nodes at the same depth are not allowed. This prevents cycles from forming and allows inference to be performed in a timely manner. More generally, one could extend the definition of $\mathcal{F}_t$ to include any node at a lower depth.

1. For node $t \in \{1, \ldots, \infty\}$
    i. Sample stick-break weights $\{v_{tj}\}|\alpha, \beta \sim \text{Beta}(\alpha, \beta)$
    ii. Sample word distribution $\phi_t|\eta \sim \text{Dirichlet}(\eta)$
2. For document $d \in \{1, 2, \ldots D\}$
    i. Sample a distribution over levels $\tau_d|a, b \sim \text{Beta}(a,b)$
    ii. Sample path $p_d \sim \{\mathcal{P}_t\}_{t=1}^{\infty}$
    iii. For word $i \in \{1, 2, \ldots, N_d\}$
        Sample level $l_{d,i} \sim \text{TruncatedDiscrete}(\tau_d)$
        Generate word $x_{d,i}|\{p_d, l_{d,i}, \Phi\} \sim \text{Multinomial}(\phi_{p_d[l_{di}]})$

---

Figure 3: Generative process for GraphLDA

Due to a lack of exchangeability, we must specify the stick-breaking order of the elements in $\mathcal{F}_t$. Note that despite the order, the elements of $\mathcal{F}_t$ always occur before the infinite set of new nodes in the stick-breaking permutation. We use a Metropolis-Hastings sampler proposed by [10] to learn the permutation of feasible nodes with the highest likelihood given the data.

## 3 Generative Process

Figure 3 shows the generative process for our proposed model, which we refer to as GraphLDA. We observe a collection of documents $d = 1 \ldots D$ where document $d$ has $N_d$ words. As discussed earlier, each node $t$ is associated with a stick-breaking prior $\mathcal{P}_t$. In addition, we associate with each node a multinomial distribution $\phi_t$ over words in the fashion of topic models.

A two-stage process is used to generate document $d$. First, a path through the graph is sampled from the stick-breaking distributions. We denote this path as $p_d$. The $i + 1$st node in the path is sampled from $\mathcal{P}_{p_{di}}(\cdot)$ which is the stick-breaking distribution at the $i$th node in the path. This process continues until an exit node is sampled. Then for each word $x_i$ a level in the path, $l_{di}$, is sampled from a truncated discrete distribution. The word $x_i$ is generated by the topic at level $l_{di}$ of the path $p_d$ which we denote as $p_d[l_{di}]$. In the case where we observe labeled documents and an initial graph structure the paths for document $d$ is restricted to end at the concept label of document $d$.

One possible option for the length distribution is a multinomial distribution over levels. We take a different approach and instead use a parametric smooth form. The motivation is to constrain the length distribution to have the same general functional form across documents (in contrast to the relatively unconstrained multinomial), but to allow the parameters of the distribution to be document-specific. We considered two simple options: Geometric and Poisson (both truncated to the number of possible levels). In initial experiments the Geometric performed better than the Poisson, so the Geometric was used in all experiments reported in this paper. If word $x_{di}$ has level $l_{di} = 0$ then the word is generated by the topic at the *last* node on the path and successive levels correspond to earlier nodes in the path. In the case of labeled documents, this matches our belief that a majority of words in the document should be assigned to the concept label itself.

## 4 Inference

We marginalize over the topic distributions $\phi_t$ and the stick-breaking weights $\{v_{tj}\}$. We use a collapsed Gibbs sampler [9] to infer the path assignment $p_d$ for each document, the level distribution parameter $\tau_d$ for each document, and the level assignment $l_{di}$ for each word. Of the five hyper-parameters in the model, inference is sensitive to the value of $\beta$ and $\eta$ so we place an Exponential prior on both and use a Metropolis-Hastings sampler to learn the best setting.

### 4.1 Sampling Paths

For each document, we must sample a path $p_d$ conditioned on all other paths $\mathbf{p}_{-d}$, the level variables, and the word tokens. We only consider paths whose length is greater than or equal to the maximum

level of the words in the document.

$$p(p_d|\mathbf{x}, \mathbf{l}, \mathbf{p}_{-d}, \boldsymbol{\tau}) \propto p(\mathbf{x}_d|\mathbf{x}_{-d}, \mathbf{l}, \mathbf{p}) \cdot p(p_d|\mathbf{p}_{-d}) \tag{1}$$

The first term in Equation 1 is the probability of all words in the document given the path $p_d$. We compute this probability by marginalizing over the topic distributions $\phi_t$:

$$p(\mathbf{x}_d|\mathbf{x}_{-d}, \mathbf{l}, \mathbf{p}) = \prod_{l=1}^{\lambda_d}\left(\prod_{v=1}^{V}\frac{\Gamma(\eta + N_{p_d[l],v})}{\Gamma(\eta + N_{p_d[l],v}^{-d})}\right) \ast \frac{\Gamma(V\eta + \sum_v N_{p_d[l],v}^{-d})}{\Gamma(V\eta + \sum_v N_{p_d[l],v})}$$

We use $\lambda_d$ to denote the length of path $p_d$. The notation $N_{p_d[l],v}$ stands for the number of times word type $v$ has been assigned to node $p_d[l]$. The superscript $-d$ means we first decrement the count $N_{p_d[l],v}$ for every word in document $d$.

The second term is the conditional probability of the path $p_d$ given all other paths $\mathbf{p}_{-d}$. We present the sampling equation under the assumption that there is a maximum number of nodes $M$ allowed at each level. We first consider the probability of sampling a single edge in the path from a node $x$ to one of its feasible nodes $\{y_1, y_2, \ldots, y_M\}$ where the node $y_1$ has the first position in the stick-breaking permutation, $y_2$ has the second position, $y_3$ the third and so on.

We denote the number of paths that have gone from $x$ to $y_i$ as $N_{(x,y_i)}$. We denote the number of paths that have gone from $x$ to a node with a strictly higher position in the stick-breaking distribution than $y_i$ as $N_{(x,>y_i)}$. That is, $N_{(x,>y_i)} = \sum_{k=i+1}^{M} N_{(x,y_k)}$. Extending this notation we denote the sum $N_{(x,y_i)} + N_{(x,>y_i)}$ as $N_{(x,\geq y_i)}$. The probability of selecting node $y_i$ is given by:

$$p(x \to y_i \mid \mathbf{p}_{-d}) = \frac{\alpha + N_{(x,y_i)}}{\alpha + \beta + N_{(x,\geq y_i)}} \prod_{r=1}^{i-1}\frac{\beta + N_{(x,>y_r)}}{\alpha + \beta + N_{(x,\geq y_r)}} \qquad \text{for } i = 1\ldots M$$

If $y_m$ is the last node with a nonzero count $N_{(x,y_m)}$ and $m << M$ it is convenient to compute the probability of transitioning to $y_i$, for $i \leq m$, and the probability of transitioning to any node higher than $y_m$. The probability of transitioning to a node higher than $y_m$ is given by

$$\sum_{k=m+1}^{M} p(x \to y_k|\mathbf{p}_{-d}) = \Delta\left[1 - \frac{\beta}{\alpha + \beta}^{M-m}\right]$$

where $\Delta = \prod_{r=1}^{m}\frac{\beta + N_{(x,>y_r)}}{\alpha + \beta + N_{(x,\geq y_r)}}$. A similar derivation can be used to compute the probability of sampling a node higher than $y_m$ when $M$ is equal to infinity. Now that we have computed the probability of a single edge, we can compute the probability of an entire path $p_d$:

$$p(p_d|\mathbf{p}_{-d}) = \prod_{j=1}^{\lambda_d} p(p_{dj} \to p_{d,j+1}|\mathbf{p}_{-d})$$

## 4.2 Sampling Levels

For the $i$th word in the $d$th document we must sample a level $l_{di}$ conditioned on all other levels $\mathbf{l}_{-di}$, the document paths, the level parameters $\boldsymbol{\tau}$, and the word tokens.

$$p(l_{di}|\mathbf{x}, \mathbf{l}_{-di}, \mathbf{p}, \boldsymbol{\tau}) = \left(\frac{\eta + N_{p_d[l_{di}],x_{di}}^{-di}}{W\eta + N_{p_d[l_{di}],\cdot}^{-di}}\right) \cdot \frac{(1 - \tau_d)^{l_{di}}\tau_d}{(1 - (1 - \tau_d)^{\lambda_d+1})}$$

The first term is the probability of word type $x_{di}$ given the topic at node $p_d[l_{di}]$. The second term is the probability of the level $l_{di}$ given the level parameter $\tau_d$.

## 4.3 Sampling $\tau$ Variables

Finally, we must sample the level distribution $\tau_d$ conditioned on the rest of the level parameters $\boldsymbol{\tau}_{-d}$, the level variables, and the word tokens.

$$p(\tau_d|\mathbf{x}, \mathbf{l}, \mathbf{p}, \boldsymbol{\tau}_{-d}) = \left(\prod_{i=1}^{N_d}\frac{(1 - \tau_d)^{l_{di}}\tau_d}{(1 - (1 - \tau_d)^{\lambda_d+1})}\right) \ast \left(\frac{\tau_d^{a-1}(1 - \tau_d)^{b-1}}{\mathbf{B}(a,b)}\right) \tag{2}$$

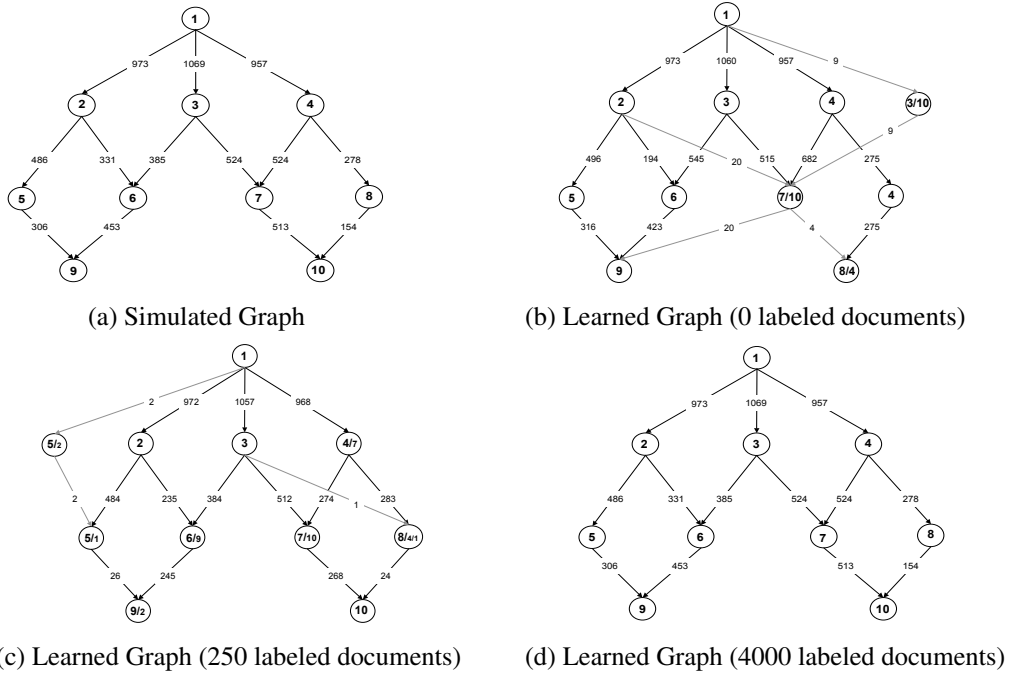

(a) Simulated Graph

(b) Learned Graph (0 labeled documents)

(c) Learned Graph (250 labeled documents)

(d) Learned Graph (4000 labeled documents)

Figure 4: Learning results with simulated data

Due to the normalization constant $(1 - (1 - \tau_d)^{\lambda_d + 1})$, Equation 2 is not a recognizable probability distribution and we must use rejection sampling. Since the first term in Equation 2 is always less than or equal to 1, the sampling distribution is dominated by a Beta$(a, b)$ distribution. According to the rejection sampling algorithm, we sample a candidate value for $\tau_d$ from Beta$(a, b)$ and either accept with probability $\prod_{i=1}^{N_d} \frac{(1 - \tau_d)^{l_{di}} \tau_d}{(1 - (1 - \tau_d)^{\lambda_d + 1})}$ or reject and sample again.

### 4.4 Metropolis Hastings for Stick-Breaking Permutations

In addition to the Gibbs sampling, we employ a Metropolis Hastings sampler presented in [10] to mix over stick-breaking permutations. Consider a node $x$ with feasible nodes $\{y_1, y_2, \ldots, y_M\}$. We sample two feasible nodes $y_i$ and $y_j$ from a uniform distribution[3]. Assume $y_i$ comes before $y_j$ in the stick-breaking distribution. Then the probability of swapping the position of nodes $y_i$ and $y_j$ is given by

$$\min\left\{1, \prod_{k=0}^{N_{(x,y_i)}-1} \frac{\alpha + \beta + N_{(x,>y_i)}^* + k}{\alpha + \beta + N_{(x,>y_j)} + k} \cdot \prod_{k=0}^{N_{(x,y_j)}-1} \frac{\alpha + \beta + N_{(x,>y_j)} + k}{\alpha + \beta + N_{(x,>y_i)}^* + k}\right\}$$

where $N_{(x,>y_i)}^* = N_{(x,>y_i)} - N_{(x,y_j)}$. See [10] for a full derivation. After every new path assignment, we propose one swap for each node in the graph.

## 5 Experiments and Results

In this section, we present experiments performed on both simulated and real text data. We compare the performance of GraphLDA against hPAM and hLDA.

### 5.1 Simulated Text Data

In this section, we illustrate how the performance of GraphLDA improves as the fraction of labeled data increases. Figure 4(a) shows a simulated concept graph with 10 nodes drawn according to the

stick-breaking generative process with parameter values $\eta = .025$, $\alpha = 10$, $\beta = 10$, $a = 2$ and $b = 5$. The vocabulary size is $1,000$ words and we generate $4,000$ documents with $250$ words each. Each edge in the graph is labeled with the number of paths that traverse it.

Figures 4(b)-(d) show the learned graph structures as the fraction of labeled data increases from 0 labeled and $4,000$ unlabeled documents to all $4,000$ documents being labeled. In addition to labeling the edges, we label each node based upon the similarity of the learned topic at the node to the topics of the original graph structure. The Gibbs sampler is initialized to a root node when there is no labeled data. With labeled data, the Gibbs sampler is initialized with the correct placement of nodes to levels. The sampler does not observe the edge structure of the graph nor the correct number of nodes at each level (i.e. the sampler may add additional nodes). With no labeled data, the sampler is unable to recover the relationship between concepts 8 and 10 (due to the relatively small number of documents that contain words from both concepts). With 250 labeled documents, the sampler is able to learn the correct placement of both nodes 8 and 10 (although the topics contain some noise).

## 5.2 Wikipedia Articles

In this section, we compare the performance of GraphLDA to hPAM and hLDA on a set of $518$ machine-learning articles taken from Wikipedia. The input to each model is only the article text. All models are restricted to learning a three-level hierarchical structure. For both GraphLDA and hPAM, the number of nodes at each level was set to $25$. For GraphLDA, the parameters were fixed at $\alpha = 1$, $a = 1$ and $b = 1$. The parameters $\beta$ and $\eta$ were initialized to $1$ and $.001$ respectively and optimized using a Metropolis Hastings sampler. We used the MALLET toolkit implementation of hPAM[4] and hLDA [11]. For hPAM, we used different settings for the topic hyperparameter $\eta = (.001, .01, .1)$. For hLDA we set $\eta = .1$ and considered $\gamma = (.1, 1, 10)$ where $\gamma$ is the smoothing parameter for the Chinese restaurant process and $\alpha = (.1, 1, 10)$ where $\alpha$ is the smoothing over levels in the graph.

All models were run for $9,000$ iterations to ensure burn-in and samples were taken every 100 iterations thereafter, for a total of $10,000$ iterations. The performance of each model was evaluated on a hold-out set consisting of $20\%$ of the articles using both empirical likelihood and the left-to-right evaluation algorithm (see Sections $4.1$ and $4.5$ of [12]) which are measures of generalization to unseen data. For both GraphLDA and hLDA we use the distribution over paths that was learned during training to compute the per-word log likelihood. For hPAM we compute the MLE estimate of the Dirichlet hyperparameters for both the distribution over super-topics and the distributions over sub-topics from the training documents. Table 5.2 shows the per-word log-likelihood for each model averaged over the ten samples. GraphLDA is competitive when computing the empirical log likelihood. We speculate that GraphLDA's lower performance in terms of left-to-right log-likelihood is due to our choice of the geometric distribution over levels (and our choice to position the geometric distribution at the last node of the path) and that a more flexible approach could result in better performance.

Table 1: Per-word log likelihood of test documents

| Model | Parameters | Empirical LL | Left-to-Right LL |
|---|---|---|---|
| GraphLDA | MH opt. | -7.10 ± .003 | -7.13 ± .009 |
| hPAM | $\eta = .1$ | -7.36 ± .013 | -6.11 ± .007 |
| | $\eta = .01$ | -7.33 ± .012 | -6.47 ± .012 |
| | $\eta = .001$ | -7.38 ± .006 | -6.71 ± .013 |
| hLDA | $\gamma = .1,\ \alpha = .1$ | -7.10 ± .004 | -6.82 ± .007 |
| | $\gamma = .1,\ \alpha = 1$ | -7.09 ± .003 | -6.86 ± .006 |
| | $\gamma = .1,\ \alpha = 10$ | -7.08 ± .003 | -6.90 ± .008 |
| | $\gamma = 1,\ \ \alpha = .1$ | -7.08 ± .003 | -6.83 ± .007 |
| | $\gamma = 1,\ \ \alpha = 1$ | -7.08 ± .002 | -6.86 ± .006 |
| | $\gamma = 1,\ \ \alpha = 10$ | -7.06 ± .003 | -6.88 ± .008 |
| | $\gamma = 10,\ \alpha = .1$ | -7.07 ± .004 | -6.81 ± .006 |
| | $\gamma = 10,\ \alpha = 1$ | -7.07 ± .003 | -6.83± .005 |
| | $\gamma = 10,\ \alpha = 10$ | -7.06 ± .003 | -6.88 ± .010 |

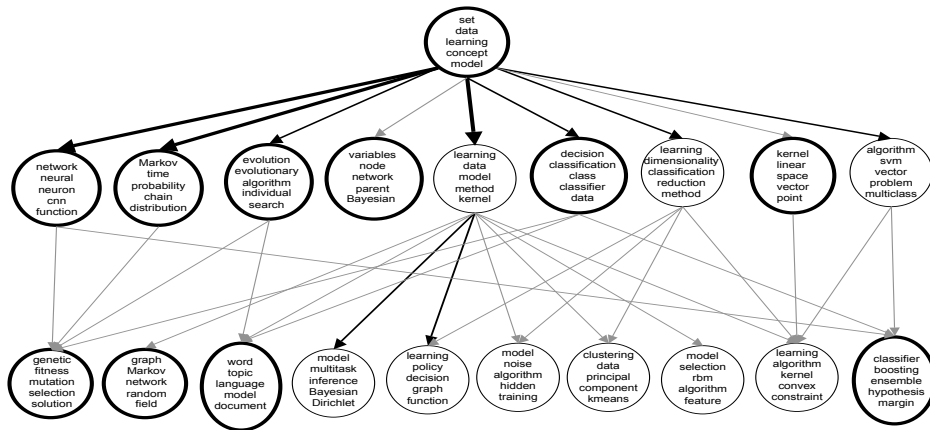

Figure 5: Wikipedia graph structure with additional machine learning abstracts. The edge widths correspond to the probability of the edge in the graph

## 5.3 Wikipedia Articles with a Graph Structure

In our final experiment we illustrate how GraphLDA can be used to update an existing category graph. We use the aforementioned 518 machine-learning Wikipedia articles, along with their category labels, to learn topic distributions for each node in Figure 1. The sampler is initialized with the correct placement of nodes and each document is initialized to a random path from the root to its category label. After 2,000 iterations, we fix the path assignments for the Wikipedia articles and introduce a new set of documents. We use a collection of 400 machine learning abstracts from the International Conference on Machine Learning (ICML). We sample paths for the new collection of documents keeping the paths from the Wikipedia articles fixed. The sampler was allowed to add new nodes to each level to explain any new concepts that occurred in the ICML text set. Figure 5 illustrates a portion of the final graph structure. The nodes in bold are the original nodes from the Wikipedia category graph. The results show that the model is capable of augmenting an existing concept graph with new concepts (e.g. clustering, support vector machines (SVMs), etc.) and learning meaningful relationships (e.g. boosting/ensembles are on the same path as the concepts for SVMs and neural networks).

## 6 Discussion and Conclusion

Motivated by the increasing availability of large-scale structured collections of documents such as Wikipedia, we have presented a flexible non-parametric Bayesian framework for learning concept graphs from text. The proposed approach can combine unlabeled data with prior knowledge in the form of labeled documents and existing graph structures. Extensions such as allowing the model to handle multiple paths per document are likely to be worth pursuing. In this paper we did not discuss scalability to large graphs which is likely to be an important issue in practice. Computing the probability of every path during sampling, where the number of graphs is a product over the number of nodes at each level, is a computational bottleneck in the current inference algorithm and will not scale. Approximate inference methods that can address this issue should be quite useful in this context.

## 7 Acknowledgements

This material is based upon work supported in part by the National Science Foundation under Award Number IIS-0083489, by a Microsoft Scholarship (AC), and by a Google Faculty Research award (PS). The authors would also like to thank Ian Porteous and Alex Ihler for useful discussions.

## Footnotes

[1] http://en.wikipedia.org/wiki/Category:Main_topic_classifications

[2] As of May 5, 2009

[3]In [10] feasible nodes are sampled from the prior probability distribution. However for small values of $\alpha$ and $\beta$ this results in extremely slow mixing.

[4]MALLET implements the "exit node" version of hPAM

# References

[1] David Blei, Andrew Ng, and Michael Jordan. Latent dirichlet allocation. *Journal of Machine Learning Research*, 3:993–1022, 2003.

[2] David M. Blei, Thomas L. Griffiths, and Michael I. Jordan. The nested chinese restaurant process and bayesian nonparametric inference of topic hierarchies. *Journal of the Acm*, 57, 2010.

[3] David Mimno, Wei Li, and Andrew McCallum. mixtures of hierarchical topics with pachinko allocation. In *Proceedings of the 21st Intl. Conf. on Machine Learning*, 2007.

[4] Wei Li, David Blei, and Andrew McCallum. Nonparametric bayes pachinko allocation. In *Proceedings of the Twenty-Third Annual Conference on Uncertainty in Artificial Intelligence (UAI-07)*, pages 243–250, 2007.

[5] Blaz Fortuna, Marko Grobelnki, and Dunja Mladenic. Ontogen: Semi-automatic ontology editor. In *Proceedings of theHuman Computer Interaction International Conference*, volume 4558, pages 309–318, 2007.

[6] S. Bloehdorn, P. Cimiano, and A. Hotho. Learning ontologies to improve text clustering and classification. In *From Data and Inf. Analysis to Know. Eng.: Proc. of the 29th Annual Conf. the German Classification Society (GfKl '05)*, volume 30 of *Studies in Classification, Data Analysis and Know. Org.*, pages 334–341. Springer, Feb. 2005.

[7] P. Cimiano, A. Hotho, and S. Staab. Learning concept hierarchies from text using formal concept analysis. *J. Artificial Intelligence Research (JAIR)*, 24:305–339, 2005.

[8] Hemant Ishwaran and Lancelot F. James. Gibbs sampling methods for stick-breaking priors. *Journal of the American Statistical Association*, 96(453):161–173, March 2001.

[9] Tom Griffiths and Mark Steyvers. Finding scientific topics. *Proceedings of the Natl. Academy of the Sciences of the U.S.A.*, 101 Suppl 1:5228–5235, 2004.

[10] Ian Porteous, Alex Ihler, Padhraic Smyth, and Max Welling. Gibbs sampling for coupled infinite mixture models in the stick-breaking representation. In *Proceedings of UAI 2006*, pages 385–392, July 2006.

[11] Andrew Kachites McCallum. Mallet: A machine learning for language toolkit. http://mallet.cs.umass.edu, 2002.

[12] Hanna M. Wallach, Iain Murray, Ruslan Salakhutdinov, and David Mimno. Evaluation methods for topic models. In *Proceedings of the 26th Intl. Conf. on Machine Learning (ICML 2009)*, 2009.

